# *VISIT*: A Neural Model of Covert Visual Attention

**Subutai Ahmad***
Siemens Research and Development,
ZFE ST SN6, Otto-Hahn Ring 6,
8000 Munich 83, Germany.
ahmad%bsun4@ztivax.siemens.com

## Abstract

Visual attention is the ability to dynamically restrict processing to a subset of the visual field. Researchers have long argued that such a mechanism is necessary to efficiently perform many intermediate level visual tasks. This paper describes *VISIT*, a novel neural network model of visual attention. The current system models the search for target objects in scenes containing multiple distractors. This is a natural task for people, it is studied extensively by psychologists, and it requires attention. The network's behavior closely matches the known psychophysical data on visual search and visual attention. *VISIT* also matches much of the physiological data on attention and provides a novel view of the functionality of a number of visual areas. This paper concentrates on the biological plausibility of the model and its relationship to the primary visual cortex, pulvinar, superior colliculus and posterior parietal areas.

## 1   INTRODUCTION

Visual attention is perhaps best understood in the context of visual search, i.e. the detection of a target object in images containing multiple distractor objects. This task requires solving the binding problem and has been extensively studied in psychology (see[16] for a review). The basic experimental finding is that a target object containing a single distinguishing feature can be detected in constant time, independent of the number of distractors. Detection based on a conjunction of features, however, takes time linear in the number of objects, implying a sequential search process (there are exceptions to this general rule). It is generally accepted

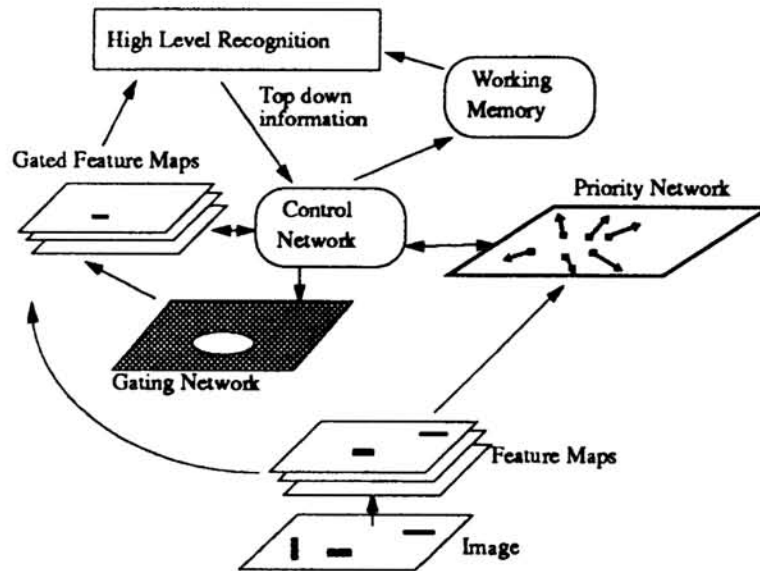

Figure 1: Overview of *VISIT*

that some form of covert attention[1] is necessary to accomplish this task. The following sections describe *VISIT*, a connectionist model of this process. The current paper concentrates on the relationships to the physiology of attention, although the psychological studies are briefly touched on. For further details on the psychological aspects see[1, 2].

## 2   OVERVIEW OF *VISIT*

We first outline the essential characteristics of *VISIT*. Figure 1 shows the basic architecture. A set of features are first computed from the image. These features are analogous to the topographic maps computed early in the visual system. There is one unit per location per feature, with each unit computing some local property of the image. Our current implementation uses four feature maps: red, blue, horizontal, and vertical. A parallel global sum of each feature map's activity is computed and is used to detect the presence of activity in individual maps.

The feature information is fed through two different systems: a gating network and a priority network. The gating network implements the focus - its function is to restrict higher level processing to a single circular region. Each gate unit receives the coordinates of a circle as input. If it is outside the circle, it turns on and inhibits corresponding locations in the gated feature maps. Thus the network can filter image properties based on an external control signal. The required computation is a simple second order weighted sum and takes two time steps[1].

The priority network ranks image locations in parallel and encodes the information in a manner suited to the updating of the focus of attention. There are three units per location in the priority map. The activity of the first unit represents the location's relevance to the current task. It receives activation from the feature maps in a local neighborhood of the image. The value of the $i$'th such unit is calculated as:

$$A_i = G( \sum_{x,y \in RF_i} \sum_{f \in F} P_f A_{fxy} ) \qquad (1)$$

where $A_{fxy}$ is the activation of the unit computing feature $f$ at location $(x, y)$. $RF_i$ denotes the receptive field of unit $i$, $P_f$ is the priority given to feature map $f$, and $G$ is a monotonically increasing function such as the sigmoid. $P_f$ is represented as the real valued activation of individual units and can be dynamically adjusted according to the task. Thus by setting $P_f$ for a particular feature to 1 and all others to 0, only objects containing that feature will influence the priority map. Section 2.1 describes a good strategy for setting $P_f$. The other two units at each location encode an "error vector", i.e. the vector difference between the units' location and center of the focus. These vectors are continually updated as the focus of attention moves around. To shift the focus to the most relevant location, the network simply adds the error vector corresponding to the highest priority unit to the activations of the units representing the focii's center. Once a location has been visited, the corresponding relevance unit is inhibited, preventing the network from continually attending to the highest priority location.

The control networks are responsible for mediating the information flow between the gating and priority networks, as well as incorporating top-down knowledge. The following section describes the part which sets the priority values for the feature maps. The rest of the networks are described in detail in [1]. Note that the control functions are fully implemented as networks of simple units and thus requires no "homunculus" to oversee the process.

## 2.1   SWIFT: A FAST SEARCH STRATEGY

The main function of SWIFT is to integrate top-down and bottom-up knowledge to efficiently guide the search process. Top down information about the target features are stored in a set of units. Let $T$ be this set of features. Since the desired object must contain all the features of $T$, any of the corresponding feature maps may be searched. Using the ability to weight feature maps differently, the network removes the influence of all but one of the features in $T$. By setting this map's priority to 1, and all others to 0, the system will effectively prune objects which do not contain this feature.SWIFT[2] To minimize search time, it should choose the feature corresponding to the smallest number of objects. Since it is difficult to count the number of objects in parallel, the network chooses the map with the minimal total activity as the one likely to contain the minimal number of objects. (If the target features are not known in advance, SWIFT chooses the minimal feature map over all features. The net effect is to always pick the most distinctive feature.)

## 2.2    RELATIONSHIP TO PSYCHOPHYSICAL DATA

The run time behavior of the system closely matches the data on human visual search. Visual attention in people is known to be very quick, taking as little as 40-80 msecs to engage. Given that cortical neurons can fire about once every 10 msecs, this leaves time for at most 8 sequential steps. In *VISIT*, unlike other implementations of attention[10], the calculation of the next location is separated from the gating process. This allows the gating to be extremely fast, requiring only 2 time steps. Iterative models, which select the most active object through lateral inhibition, require time proportional to the distance in pixels between maximally separated objects. These models are not consistent with the 80msecs time requirement.

During visual search, *SWIFT* always searches the minimal feature map. The critical variable that determines search time is $M$, the number of objects in the minimal feature map. Search time will be linear in $M$. It can be shown that *VISIT* plus *SWIFT* is consistent with all of Treisman's original experiments including single feature search, conjunctive search, 2:1 slope ratios, search asymmetries, and illusory conjuncts[16], as well as the exceptions reported in[5, 14]. With an assumption about the features that are coded (consistent with current physiological knowledge), the results in[7, 11] can also be modeled. (This is described in more detail in [2]).

# 3    PHYSIOLOGY OF VISUAL ATTENTION

The above sections have described the general architecture of *VISIT*. There is a fairly strong correspondence between the modules in *VISIT* and the various visual areas involved in attention. The rest of the paper discusses these relationships.

## 3.1    TOPOGRAPHIC FEATURE MAPS

Each of the early visual areas, $LGN$, $V1$, and $V2$, form several topographic maps of retinal activity. In $V1$ alone there are a thousand times as many neurons as there are fibers in the optic nerve, enough to form several hundred feature maps. There is a diverse list of features thought to be computed in these areas, including orientations, colors, spatial frequencies, motion, etc.[6]. These areas are analogous to the set of early feature maps computed in *VISIT*.

In *VISIT* there are actually two separate sets of feature maps: early features computed directly from the image and gated feature maps. It might seem inefficient to have two copies of the same features. An alternate possibility is to directly inhibit the early feature maps themselves, and so eliminate the need for two sets. However, in a focused state, such a network would be unable to make global decisions based on the features. With the configuration described above, at some hardware cost, the network can efficiently access both local and global information simultaneously. *SWIFT* relies on this ability to efficiently carry out visual search.

There is evidence for a similar setup in the human visual system. Although people have actively searched, no local attentional effects have been found in the early feature maps. (Only *global* effects, such as an overall increase in firing rate, have been noticed.) The above reasoning provides a possible computational explanation of this phenomenon.

A natural question to ask is: what is the best set of features? For fast visual search, if *SWIFT* is used as a constraint, then we want the set of features that minimize $M$ over all possible images and target objects, i.e. the features that best discriminate objects. It is easy to see that the optimal set of features should be maximally uncorrelated with a near uniform distribution of feature values. Extracting the principal components of the distribution of images gives us exactly those features. It is well known that a single Hebb neuron extracts the largest principal component; sets of such neurons can be connected to select successively smaller components. Moreover, as some researchers have demonstrated, simple Hebbian learning can lead to features that look very similar to the features in visual cortex (see [3] for a review). If the early features in visual cortex do in fact represent principal components, then *SWIFT* is a simple strategy that takes advantage of it.

### 3.2  THE PULVINAR

Contrary to the early visual system, local attentional effects have been discovered in the pulvinar. Recordings of cells in the lateral pulvinar of awake, behaving monkeys have demonstrated a spatially localized enhancement effect tied to selective attention[17]. Given this property it is tempting to pinpoint the pulvinar as the locus of the gated feature maps.

The general connectivity patterns provide some support for this hypothesis. The pulvinar is located in the dorsal part of the thalamus and is strongly connected to just about every visual area including $LGN$, $V1$, $V2$, superior colliculus, the frontal eye fields, and posterior parietal cortex. The projections are topography preserving and non-overlapping. As a result, the pulvinar contains several high-resolution maps of visual space, possibly one map for each one in primary visual cortex. In addition, there is a thin sheet of neurons around the pulvinar, the reticular complex, with exclusively inhibitory connections to the neurons within [4]. This is exactly the structure necessary to implement *VISIT*'s gating system.

There are other clues which also point to the thalamus as the gating system. Human patients with thalamic lesions have difficulty engaging attention and inhibiting crosstalk from other locations. Lesioned monkeys give slower responses when competing events are present in the visual field[12].

The hypothesis can be tested by further experiments. In particular, if a map in the pulvinar corresponding to a particular cortical area is damaged, then there should be a corresponding deficit in the ability to bind those specific features in the presence of distractors. In the absence of distractors, the performance should remain unchanged.

### 3.3  SUPERIOR COLLICULUS

The $SC$ is involved in both the generation of eye saccades[15] and possibly with covert attention[12]. It is probably also involved in the integration of location information from various different modalities. Like the pulvinar, the superior colliculus ($SC$) is a structure with converging inputs from several different modalities including visual, auditory, and somatosensory[15]. The superior colliculus contains a representation similar to *VISIT*'s error maps for eye saccades[15]. At each location,

groups of neurons represent the vector in motor coordinates required to shift the eye to that spot. In [13] the authors studied patients with a particular form of Parkinson's disease where the $SC$ is damaged. These patients are able to make horizontal, but not vertical eye saccades. The experiments showed that although the patients were still able to move their covert attention in both the horizontal and vertical directions, the speed of orienting in the vertical direction was much slower. In addition [12] mentions that patients with this damage shift attention to previously attended locations as readily as new ones, suggesting a deficit in the mechanism that inhibits previously attended locations.

These findings are consistent with the priority map in *VISIT*. A first guess would identify the superior colliculus as the priority map, however this is probably inaccurate. More recent evidence suggests that the $SC$ might be involved only in bottom-up shifts of attention (induced by exogenous stimuli as opposed to endogenous control signals) (Rafal, personal communication). There is also evidence that the frontal eye fields ($FEF$) are involved in saccade generation in a manner similar to the superior colliculus, particularly for saccades to complex stimuli[17]. The role of the $FEF$ in covert attention is currently unknown.

### 3.4   POSTERIOR PARIETAL AREAS

The posterior paretal cortex $PP$ may provide an answer. One hypothesis that is consistent with the data is that there are several different priority maps, for bottom-up and top-down stimuli. The top-down maps exist within $PP$, whereas the bottom-up maps exist in $SC$ and possibly $FEF$. $PP$ receives a significant projection from superior colliculus and may be involved in the production of voluntary eye saccades[17]. Experiments suggest that it is also involved in covert shifts of attention. There is evidence that neurons in $PP$ increase their firing rate when in a state of attentive fixation[9]. Damage to $PP$ leads to deficits in the ability to disengage covert attention away from a target[12]. In the context of eye saccades, there exist neurons in $PP$ that fire about 55 msecs before an actual saccade. These results suggest that the control structure and the aspects of the network that integrate priority information from the various modules might also reside within $PP$.

## 4   DISCUSSION AND CONCLUSIONS

The above relationships between *VISIT* and the brain provides a coherent picture of the functionality of the visual areas. The literature is consistent with having the $LGN$, $V1$, and $V2$ as the early feature maps, the pulvinar as a gating system, the superior colliculus, and frontal eye fields, as a bottom-up priority map, and posterior parietal cortex as the locus of a higher level priority map as well as the the control networks. Figure 2 displays the various visual areas together with their proposed functional relationships.

In [12] the authors suggest that neurons in parietal lobe disengage attention from the present focus, those in superior colliculus shift attention to the target, and neurons in pulvinar engage attention on it. This hypothesis looks at the time course of an attentional shift (disengage, move, engage) and assigns three different areas to

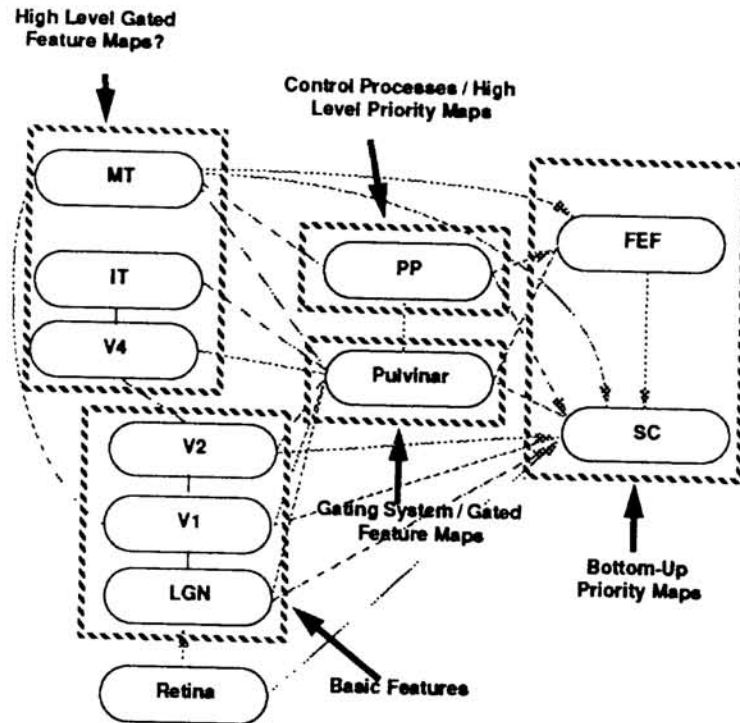

Figure 2: Proposed functionality of various visual areas. Lines denote major pathways. Those connections without arrows are known to be bi-directional.

the three different intervals within that temporal sequence. In *VISIT*, these three correspond to a single operation (add a new update vector to the current location) and a single module (the control network). Instead, the emphasis is on assigning different computational responsibilities to the various modules. Each module operates continuously but is involved in a different computation. While the gating network is being updated to a new location, the priority network and portions of the control network are continuously updating the priorities.

The model doesn't yet explain the findings in [8] where neurons in $V4$ exhibited a localized attentional response, but only if the stimuli were within the receptive fields. However, these neurons have relatively large receptive fields and are known to code for fairly high-level features. It is possible that this corresponds to a different form of attention working at a much higher level.

By no means is *VISIT* intended to be a detailed physiological model of attention. Precise modeling of even a single neuron can require significant computational resources. There are many physiological details that are not incorporated. However, at the macro level there are interesting relationships between the individual modules in *VISIT* and the known functionality of the different areas. The advantage of an implemented computational model such as *VISIT* is that it allows us to examine the underlying computations involved and hopefully better understand the underlying processes.

## Footnotes

*Thanks to Steve Omohundro, Anne Treisman, Joe Malpeli, and Bill Baird for enlightening discussions. Much of this research was conducted at the International Computer Science Institute, Berkeley, CA.

[1] *Covert attention* refers to the ability to concentrate processing on a single image region without any overt actions such as eye movements.

[2]Hence the name SWIFT: Search WIth Features Thrown out.

# References

[1] S. Ahmad. *VISIT: An Efficient Computational Model of Human Visual Attention.* PhD thesis, University of Illinois at Urbana-Champaign, Champaign, IL, September 1991. Also TR-91-049, International Computer Science Institute, Berkeley, CA.

[2] S. Ahmad and S. Omohundro. Efficient visual search: A connectionist solution. In *13th Annual Conference of the Cognitive Science Society*, Chicago, IL, August 1991.

[3] S. Becker. Unsupervised learning procedures for neural networks. *International Journal of Neural Systems*, 12, 1991.

[4] F. Crick. Function of the thalamic reticular complex: the searchlight hypothesis. In *National Academy of Sciences*, volume 81, pages 4586–4590, 1984.

[5] H.E. Egeth, R.A. Virzi, and H. Garbart. Searching for conjunctively defined targets. *Journal of Experimental Psychology: Human Perception and Performance*, 10(1):32–39, 1984.

[6] D. Van Essen and C. H. Anderson. Information processing strategies and pathways in the primate retina and visual cortex. In S.F. Zornetzer, J.L. Davis, and C. Lau, editors, *An Introduction to Neural and Electronic Networks*. Academic Press, 1990.

[7] P. McLeod, J. Driver, and J. Crisp. Visual search for a conjunction of movement and form is parallel. *Nature*, 332:154–155, 1988.

[8] J. Moran and R. Desimone. Selective attention gates visual processing in the extrastriate cortex. *Science*, 229, March 1985.

[9] V.B. Mountcastle, R.A. Anderson, and B.C. Motter. The influence of attention fixation upon the excitability of the light-sensitive neurons of the posterior parietal cortex. *The Journal of Neuroscience*, 1(11):1218–1235, 1981.

[10] M. Mozer. *The Perception of Multiple Objects: A Connectionist Approach.* MIT Press, Cambridge, MA, 1991.

[11] K. Nakayama and G. Silverman. Serial and parallel processing of visual feature conjunctions. *Nature*, 320:264–265, 1986.

[12] M.I. Posner and S.E. Petersen. The attention system of the human brain. *Annual Review of Neuroscience*, 13:25–42, 1990.

[13] M.I. Posner, J.A. Walker, and R.D. Rafal. Effects of parietal injury on covert orienting of attention. *The Journal of Neuroscience*, 4(7):1863–1874, 1982.

[14] P.T. Quinlan and G.W. Humphreys. Visual search for targets defined by combinations of color, shape, and size: An examination of the task constraints of feature and conjunction searches. *Perception & Psychophysics*, 41:455–472, 1987.

[15] D. L. Sparks. Translation of sensory signals into commands for control of saccadic eye movements: Role of primate superior colliculus. *Physiological Reviews*, 66(1), 1986.

[16] A. Treisman. Features and objects: The Fourteenth Bartlett Memorial Lecture. *The Quarterly Journal of Experimental Psychology*, 40A(2), 1988.

[17] R.H. Wurtz and M.E. Goldberg, editors. *The Neurobiology of Saccadic Eye Movements.* Elsevier, New York, 1989.